# Explaining human multiple object tracking as resource-constrained approximate inference in a dynamic probabilistic model

**Edward Vul, Michael C. Frank, and Joshua B. Tenenbaum**
Department of Brain and Cognitive Sciences
Massachusetts Institute of Technology
Cambridge, MA 02138
{evul, mcfrank, jbt}@mit.edu

**George Alvarez**
Department of Psychology
Harvard University
Cambridge, MA 02138
alvarez@wjh.harvard.edu

## Abstract

Multiple object tracking is a task commonly used to investigate the architecture of human visual attention. Human participants show a distinctive pattern of successes and failures in tracking experiments that is often attributed to limits on an object system, a tracking module, or other specialized cognitive structures. Here we use a computational analysis of the task of object tracking to ask which human failures arise from cognitive limitations and which are consequences of inevitable perceptual uncertainty in the tracking task. We find that many human performance phenomena, measured through novel behavioral experiments, are naturally produced by the operation of our ideal observer model (a Rao-Blackwelized particle filter). The tradeoff between the speed and number of objects being tracked, however, can only arise from the allocation of a flexible cognitive resource, which can be formalized as either memory or attention.

## 1  Introduction

Since William James first described the phenomenology of attention [11], psychologists have been struggling to specify the cognitive architecture of this process, how it is limited, and how it helps information processing. The study of visual attention specifically has benefited from rich, simple paradigms, and of these multiple object tracking (MOT) [16] has recently gained substantial popularity. In a typical MOT task (Figure 1), subjects see a number of objects, typically colorless circles, moving onscreen. Some subset of the objects are marked as targets before the trial begins, but during the trial all objects turn to a uniform color and move haphazardly for several seconds. The task is to keep track of which objects were marked as targets at the start of the trial so that they can be identified at the end of the trial when the objects stop moving.

The pattern of results from MOT experiments is complicated. Participants can only track a finite number of objects [16], but more objects can be tracked when they move slower [1], suggesting a limit on attentional speed. If objects are moved far apart in the visual field, however, they can be tracked at high speeds, suggesting that spatial crowding also limits tracking [9]. When tracking, participants seem to maintain information about the velocity of objects [19] and this information is sometimes helpful in tracking [8]. More frequently, however, velocity is not used to track, suggesting limitations on the kinds of information available to the tracking system [13]. Finally, although participants can track objects using features like color and orientation [3], some features seem to hurt tracking [15], and tracking is primarily considered to be a spatial phenomenon. These results and others have left researchers puzzled: What limits tracking performance?

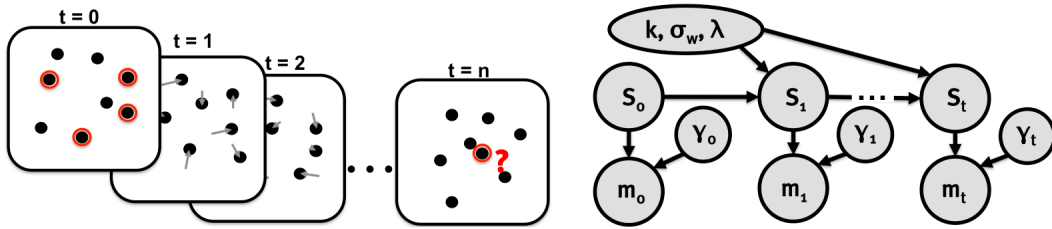

Figure 1: *Left:* A typical multiple object tracking experiment. *Right:* The generative model underlying our probabilistic tracker (see text for details).

Proposed limitations on MOT performance may be characterized along the dimensions of discreteness and flexibility. A proposal positing fixed number of slots (each holding one object) describes a discrete limitation, while proposals positing limits on attention speed or resolution are more continuous. Attention and working memory are canonical examples of flexible limitations: Based on the task, we decide where to attend and what to remember. Such cognitive limitations may be framed either as a discrete number of slots or as a continuous resource. In contrast, visual acuity and noise in velocity perception are low-level, task-independent limitations: Regardless of the task we are doing, the resolution of our retina is limited and our motion-discrimination thresholds are stable. Such perceptual limitations tend only to be continuous.

We aim to determine which MOT effects reflect perceptual, task-independent uncertainty, and which reflect flexible, cognitive limitations. Our approach is to describe the minimal computations that an ideal observer must undertake to track objects and combine available information. To the extent that an effect is not naturally explained at the computational level given only perceptual sources of uncertainty, it is more likely to reflect flexible cognitive limitations.

We propose that humans track objects in a manner consistent with the Bayesian multi-target tracking framework common in computer vision [10, 18]. We implement a variant of this tracking model using Rao-Blackwellized particle filtering and show how it can be easily adapted for a wide range of MOT experiments. This unifying model allows us to design novel experiments that interpolate between seemingly disparate phenomena. We argue that, since the effects of speed, spacing, and features arise naturally in an ideal observer with no limits on attention, memory, or number of objects that can be tracked, these phenomena can be explained by optimal object tracking given low-level, perceptual sources of uncertainty. We identify a subset of MOT phenomena that must reflect flexible cognitive resources, however: effects that manipulate the number of objects that can be tracked. To account for tradeoffs between object speed and number, a task-dependent resource constraint must be added to our model. This constraint can be interpreted as either limited attentional resolution or limited short term memory.

## 2   Optimal multiple object tracking

To track objects in a typical MOT experiment (Figure 1), at each point in time the observer must determine which of many observed objects corresponds to which of the objects that were present in the display in the last frame. Here we will formalize this procedure using a classical tracking algorithm in computer vision[10, 18].

### 2.1   Dynamics

Object tracking requires some assumptions about how objects evolve over time. Since there is no consensus on how to generate object tracking displays in the visual attention literature, we will assume simple linear dynamics, which can approximate prior experimental manipulations. Specifically, we assume that the true state of the world $S_t$ contains information about each object being tracked ($i$): to start we consider objects defined by position ($x_t(i)$) and velocity ($v_t(i)$), but we will later consider tracking objects through more complicated feature-spaces. Although we refer to position and velocity, the state actually contains two position and two velocity dimensions: one of each for x and y.

$S_t$ evolves according to linear dynamics with noise. Position and velocity for x and y evolve independently according to an Ornstein-Uhlenbeck (mean-reverting) process, which can be thought of as Brownian motion on a spring, and can be most clearly spelled out as a series of equations:

$$\begin{aligned} x_t &= x_{t-1} + v_t, \\ v_t &= \lambda v_{t-1} - k x_{t-1} + w_t, \\ w_t &\sim N(0, \sigma_w) \end{aligned} \tag{1}$$

where $x$ and $v$ are the position and velocity at time $t$; $\lambda$ is an inertia parameter constrained to be between 0 and 1; $k$ is a spring constant which produces the mean-reverting properties of the dynamics; and $w_t$ is random acceleration noise added at each time point which is distributed as a zero-mean Gaussian with standard deviation $\sigma_w$.

In two dimensions, this stochastic process describes a randomly moving cloud of objects; the spring constant assures that the objects will not drift off to infinity, and the friction parameter assures that they will not accelerate to infinity. Within the range of parameters we consider, this process converges to a stable distribution of positions and velocities both of which will be normally distributed around zero. We can solve for the standard deviations for position ($\sigma_x$) and velocity ($\sigma_v$), by assuming that the expected values of $\sigma_x$, $\sigma_v$ and their covariance will not change through an update step; thus obtaining:

$$\sigma_x = \sqrt{\frac{(1+\lambda)\sigma_w^2}{k(\lambda-1)(2\lambda-k-2)}} \text{ , and } \sigma_v = \sqrt{\frac{2\sigma_w^2}{k(\lambda-1)(2\lambda-k+2)}}, \tag{2}$$

respectively. Because these terms are familiar in the human multiple object tracking literature, for the rest of this paper we will describe the dynamics in terms of the spatial extent of the cloud of moving dots ($\sigma_x$), the standard deviation of the velocity distribution ($\sigma_v$), and the inertia parameter ($\lambda$; solving for $k$ and $\sigma_w$ to generate dynamics and track objects).

## 2.2 Probabilistic model

The goal of an object tracking model is to track the set of $n$ objects in $S$ over a fixed period from $t_0$ to $t_m$. For our model, we assume observations ($m_t$) at each time $t$ are noisy measurements of the true state of the world at that time ($S_t$). In other words, our tracking model is a stripped-down simplification of tracking models commonly used in computer vision because we do not track from *noisy images*, but instead, from extracted position and velocity estimates. The observer must estimate $S_t$ based on the current, and previous measurements, thus obtaining $\hat{S}_t$. However, this task is complicated by the fact that the observer obtains an unlabeled bag of observations ($m_t$), and does not know which observations correspond to which objects in the previous state estimate $\hat{S}_{t-1}$. Thus, the observer must not only estimate $S_t$, but must also determine the data assignment of observations to objects — which can be described by a permutation vector $\gamma_t$.

Since we assume independent linear dynamics for each individual object, then conditioned on $\gamma$, we can track each individual object via a Kalman filter. That is, what is a series of unlabeled bags of observations when data assignments were unknown, becomes a set of individuated time-series — one for each object — in which each point in time contains only a single observation when conditioned on the data assignment. The Kalman filter will be updated via transition matrix $A$, according to $S_t = A S_{t-1} + W_t$, and state perturbations $W$ are distributed with covariance $Q$ ($A$ and $Q$ can be derived straight-forwardly from the dynamics in Eq. 1; see Supplementary Materials).

Inference about both the state estimate and the data assignment can proceed by predicting the current location for each object, which will be a multivariate normal distribution with mean predicted state $\hat{S}_{t|t-1} = A\hat{S}_{t-1}$ and predicted estimate covariance $G_{t|t-1} = AG_{t-1}A' + Q$. From these predictions, we can define the probability of a particular data assignment permutation vector as:

$$P(\gamma_t | S_t, G_t, M_t) = \prod_i P(\gamma_t(i) | \hat{S}_{t|t-1}(i), G_{t|t-1}(i), M_t(i)), \text{ where}$$

$$P(\gamma i | \hat{S}_{t|t-1}(i), G_{t|t-1}(i)) = N(m_t(\gamma(i)); \hat{S}_{t|t-1}(i), G_{t|t-1}(i) + M_t(\gamma(i))) \tag{3}$$

where the posterior probability of a particular $\gamma$ value is determined by the Gaussian probability density, and $M_t(j)$ is the covariance of measurement noise for $m_t(j)$. Because an observation can

arise from only one object, mutual exclusivity is built into this conditional probability distribution — this complication makes analytical solutions impossible, and the data assignment vector, $\gamma$, must be sampled. However, given an estimate of $\gamma$, an estimate of the current state of the object is given by the Kalman state update ([12]; see Supplementary Materials).

## 2.3 Inference

To infer the state of the tracking model described above, we must sample the data-association vector, $\gamma$, and then the rest of the tracking may proceed analytically. Thus, we implement a Rao-Blackwelized particle filter [6]: each particle corresponds to one sampled $\gamma$ vector and contains the analytically computed state estimates for each of the objects, conditioned on that sampled $\gamma$ vector. Thus, taken together, the particles used for tracking (in our case we use 50, but see Section 3.4 for discussion) approximate the joint probability distribution over $\gamma$ and $S$.

In practice, we sample $\gamma$ with the following iterative procedure. First, we sample each component of $\gamma$ independently of all other $\gamma$ components (as in PMHT [18]). Then, if the resulting $\gamma$ vector contains conflicts that violate the mutual exclusivity of data assignments, a subset of $\gamma$ is resampled. If this resampling procedure fails to come up with an assignment vector that meets the mutual exclusivity, we compute the combinatoric expansion of the permutation of the conflicted subset of $\gamma$ and sample assignments within that subset from the combinatoric space. This procedure is very fast when tracking is easy, but can slow down when tracking is hard and the combinatoric expansion is necessary.

## 2.4 Perceptual uncertainty

In order to determine the limits on optimal tracking in our model, we must know what information human observers have access to. We assume that observers know the summary statistics of the cloud of moving dots (their spatial extent, given by $\sigma_x$, and their velocity distribution, $\sigma_v$). We also start with the assumption that they know the inertia parameter ($\lambda$; however, this assumption will be questioned in section 3.2). Given a perfect measurement of $\sigma_x$, $\sigma_v$, and $\lambda$, observers will thus know the dynamics by which the objects evolve.

We must also specify the low-level, task-independent noise for human observers. We assume that noise in observing the positions of objects ($\sigma_{mx}$) is given by previously published eccentricity scaling parameters, $\sigma_{mx}(x) = c(1 + 0.42x)$ (from [5]), where $c$ is uncertainty in position. We use $c = 0.08$ (standard deviation in degrees visual angle) throughout this paper. We also assume that observations of speed are corrupted by Weber-scaled noise with some irreducible uncertainty ($a$): $\sigma_{mv}(v) = a + bv$, setting $a = 0.01$ and $b = 0.05$ ($b$ is the weber fraction as measured in [17]).

# 3 Results

## 3.1 Tracking through space

When objects move faster, tracking them is harder [1], suggesting to researchers that an attentional speed limit may be limiting tracking. However, when objects cover a wider area of space (when they move on a whole field display), they can be tracked more easily at a given speed, suggesting that crowding rather than speed is the limiting factor [9].

Both of these effects are predicted by our model: both the speed and spatial separation of objects alter the uncertainty inherent in the tracking task. When objects move faster (greater $\sigma_v$) predictions about about where objects will be on the next time-step have greater uncertainty: the covariance of the predicted state ($G_{t|t-1}$) has greater entropy and inference about which observation arose from which object ($\gamma$) is less certain and more prone to errors. Additionally, even at a given speed and inertia, when the spatial extent ($\sigma_x$) is smaller, objects are closer together. Even given a fixed uncertainty about where in space an object will end up, the odds of another object appearing therein is greater, again limiting our ability to infer $\gamma$. Thus, both increasing velocity variance and decreasing spatial variance will make tracking harder, and to achieve a particular level of performance the two must trade off.

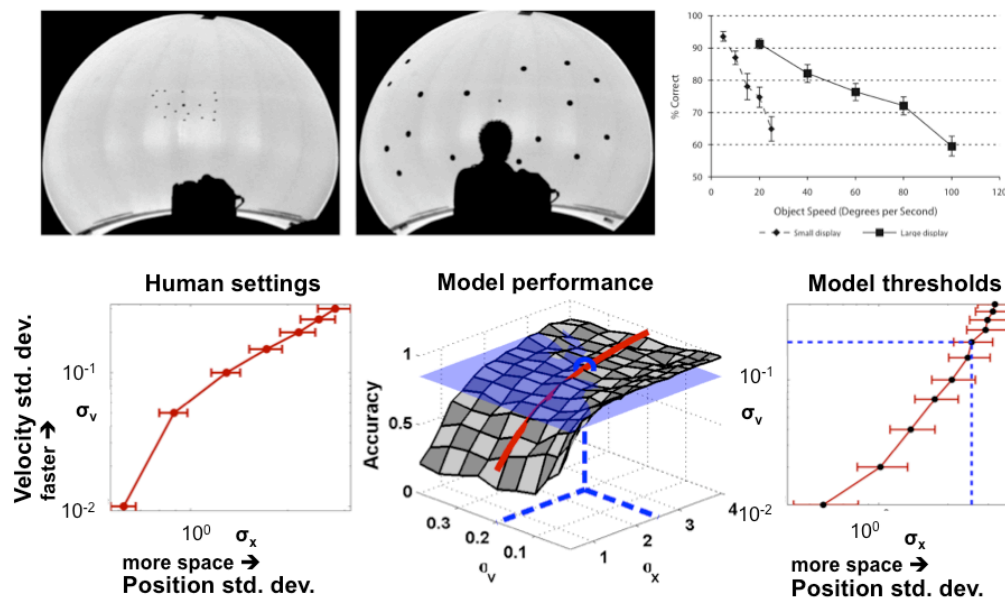

Figure 2: *Top:* Stimuli and data from [9] — when objects are tracked over the whole visual field, they can move at greater speed to achieve a particular level of accuracy. *Bottom-Left:* Our own experimental data in which subjects set a "comfortable" spacing for tracking 3 of 6 objects at a particular speed. *Bottom-Middle:* Model accuracy for tracking 3 of 6 objects as a function of speed and spacing. *Bottom-Right:* Model "settings" — (85% accuracy) threshold spacing for a particular speed. See text for details.

We show the speed-space tradeoff in both people and our ideal tracking model. We asked 10 human observers to track 3 of 6 objects moving according to the dynamics described earlier. Their goal was to adjust the difficulty of the tracking task so that they could track the objects for 5 seconds. We told them that sometimes tracking would be too hard and sometimes too easy, and they could adjust the difficulty by hitting one button to make the task easier and another button to make it harder.[1] Making the task easier or harder amounted to moving the objects farther apart or closer together by adjusting $\sigma_x$ of the dynamics, while the speed ($\sigma_v$) stayed constant. We parametrically varied $\sigma_v$ between 0.01 and 0.4, and could thus obtain an *iso-difficulty* curve for people making settings of $\sigma_x$ as a function of $\sigma_v$ (2).

To elicit predictions from our model on this task we simulated 5 second trials where the model had to track 3 of 6 objects, and measured accuracy across 15 spacing intervals ($\sigma_x$ between 0.5 and 4.0 degrees visual angle), crossed with 11 speeds ($\sigma_v$ between 0.01 and 0.4). At each point in this speed-space grid, we simulated 250 trials, to measure mean tracking accuracy for the model. The resulting accuracy surface is shown in Figure 2 — an apparent tradeoff can be seen, when objects move faster, they must be farther apart to achieve the same level of accuracy as when they move slower.

To make the model generate thresholds of $\sigma_x$ for a particular $\sigma_v$, as we had human subjects do, we fit psychometric functions to each cross-section through the accuracy surface, and used the psychometric function to predict settings that would achieve a particular level of accuracy (one such psychometric function is shown in red on the surface in Figure2).[2] The plot in Figure 2 shows the model setting for the 0.85 accuracy mark; the upper and lower error bounds represent the settings to achieve an accuracy of 0.8 and 0.9, respectively (in subsequent plots we show only the 0.85 threshold for simplicity). As in the human performance, there is a continuous tradeoff: when objects are faster, spacing must be wider to achieve the same level of difficulty.

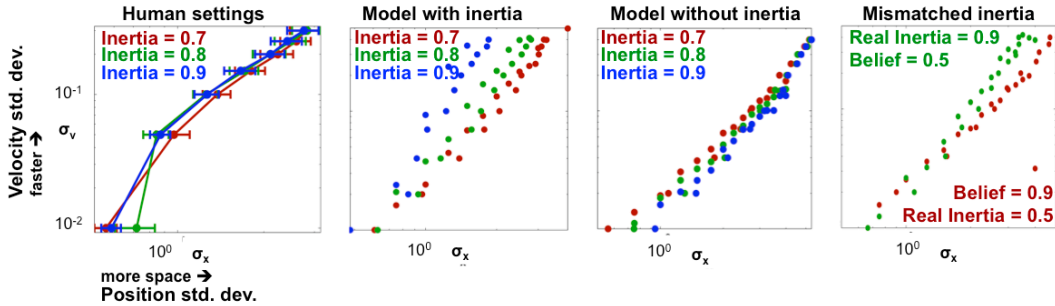

Figure 3: *Left:* Human speed-space tradeoff settings do not vary for different physical inertias. *Middle panels:* This is the case for the ideal model with no knowledge of inertia, but not so for the ideal model with perfect knowledge of inertia. *Right:* This may be the case because it is safer to assume a lower inertia: tracking is worse if inertia is assumed to be higher than it is (red) than vice versa (green).

## 3.2 Inertia

It is disputed whether human observers use velocity to track[13]. Nonetheless, it is clear that adults, and even babies, know something about object velocity [19]. The model we propose can reconcile these conflicting findings.

In our model, knowing object velocity means having an accurate $\sigma_v$ term for the object: an estimate of how much distance it might cover in a particular time step. Using velocity trajectories to make predictions about future states also requires that people know the inertia term. Thus, the degree to which trajectories are used to track is a question about the inertia parameter ($\lambda$) that best matches human performance. Thus far we have assumed that people know $\lambda$ perfectly and use it to predict future states, but this need not be the case. Indeed, while the two other parameters of the dynamics — the spatial extent ($\sigma_x$) and velocity distribution ($\sigma_v$) — may be estimated quickly and efficiently from a brief observation of the tracking display, inertia is more difficult to estimate. Thus, observers may be more uncertain about the inertia, and may be more likely to guess it incorrectly. (Under our model, a guess of $\lambda = 0$ corresponds to tracking without any velocity information.)

We ran an experiment to assess what inertia parameter best fits human observers. We asked subjects to set iso-difficulty contours as a function of the underlying inertia ($\lambda$) parameter, by using the same difficulty-setting procedure described earlier. An ideal observer who knows the inertia perfectly will greatly benefit from displays with high inertia in which uncertainty will be low, and will be able to track with the same level of accuracy at greater speeds given a particular spacing. However, if inertia is incorrectly assumed to be zero, high- and low-inertia iso-difficulty contours will be quite similar (Figure 3). We find that in human observers, iso-difficulty contours for $\lambda = 0.7$, $\lambda = 0.8$, and $\lambda = 0.9$, are remarkably similar — consistent with observers assuming a single, low, inertia term.

Although these results corroborate previous findings that human observers do not seem to use trajectories to track, there is evidence that sometime people *do* use trajectories. These variations in observers' assumptions about inertia may be attributable to two factors. First, most MOT experiments including rather sudden changes in velocity from objects bouncing off the walls or simply as a function of their underlying dynamics. Second, under uncertainty about the inertia underlying a particular display, an observer is better off underestimating rather than overestimating. Figure 3 shows the decrement in performance as a function of a mismatch of the observers' assumed inertia to that of the tracking display.

## 3.3 Tracking through feature space

In addition to tracking through space, observers can also track objects through feature domains. For example, experimental participants can track two spatially superimposed gratings based on their slowly varying colors, orientations or spatial frequencies [3].

We can modify our model to track in feature space by adding new dimensions corresponding to the features being tracked. Linear feature dimensions like the log of spatial frequency can be treated exactly like position and velocity. Circular features like hue angle and orientation require a slight

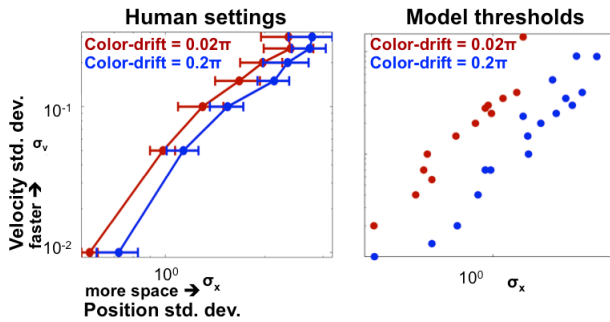

Figure 4: *Left:* When object color drifts more slowly over time (lower $\sigma_c$), people can track objects more effectively. *Right:* Our tracking model does so as well (observation noise for color $\sigma_{mc}$ in the model was set to $0.02\pi$)

modification: we pre-process the state estimates and observations via modulus to preserve their circular relationship and the linear the Kalman update. With this modification, the linear Kalman state update can operate on circular variables, and our basic tracking model can track colored objects with a high level of accuracy when they are superimposed ($\sigma_x = \sigma_v = 0$, Figure 4).

We additionally tested the novel prediction from our model that human observers can combine the information available from space and features for tracking. Nine human observers made iso-difficulty settings as described above; however, this time each object had a color and we varied the color drift rate ($\sigma_c$) on hue angle. Figure 4 shows subjects' settings of $\sigma_x$ as a function of $\sigma_v$ and $\sigma_c$. When color changes slowly, observers can track objects in a smaller space at a given velocity. Figure 4 also shows that the pattern of thresholds from the model in the same task match those of the experimental participants. Thus, not only can human observers track objects in feature space, they can combine both spatial location and featural information, and additional information in the feature domain allows people to track successfully with less spatial information, as argued by [7].

## 3.4 Cognitive limitations

Thus far we have shown that many human failures in multiple object tracking do not reflect cognitive limitations on tracking, but are instead a consequence of the structure of the task and the limits on available perceptual information. However, a limit on the number of objects that may be tracked [16] cannot be accounted for in this way. Observers can more easily track 4 of 16 objects at a higher speed than 8 of 16 objects (Figure 5), even though the stimulus presentation is identical in both cases [1]. Thus, this limitation must be a consequence of uncertainty that may be modulated by task — a flexible resource [2].

Within our model, there are two plausible alternatives for what such a limited resource may be: visual attention, which improves the fidelity of measurements; or memory, which enables more or less noiseless propagation of state estimates through time[3]. In both cases, when more objects are tracked, less of the resource is available for each object, resulting in an increase of noise and uncertainty. At a superficial level, both memory and attention resources amount to a limited amount of gain to be used to reduce noise. Given the linear Kalman filtering computation we have proposed as underlying tracking, equal magnitude noise in either will have the same effects. Thus, to avoid the complexities inherent in allocating attention to space, we will consider memory limitations, but this resource limitation can be thought of as "attention gain" as well (though some of our work suggests that memory may be a more appropriate interpretation).

We must decide on a linking function between the covariance $U$ of the memory noise, and the number of objects tracked. It is natural to propose that covariance scales positively with the number of objects tracked – that is $U$ for $n$ objects would be equal to $U_n = U_1 n$. This expression captures the idea that task modulated noise should follow the $\sigma \propto \sqrt{n}$ rule, as would be the case if the state for a given object were stored or measured with a finite number of samples. With more samples,

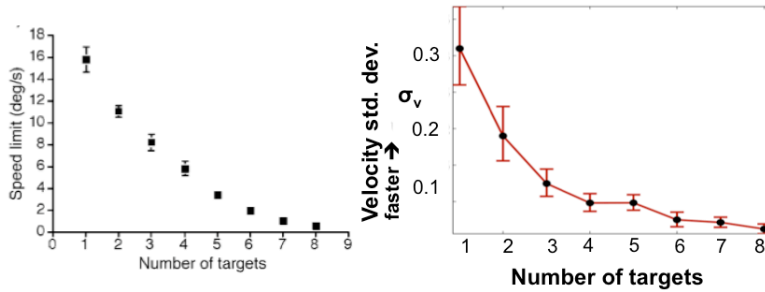

Figure 5: *Left:* When more objects are tracked (out of 16) they must move at a slower speed to reach a particular level of accuracy [1]. *Right:* Our model exhibits this effect only if task-dependent uncertainty is introduced (see text).

precision would increase; however, because the number of samples available is fixed at $c$, the number of samples per object would be $c/n$, giving rise to the scaling rule described above.

In Figure 5 we add such a noise-term to our model and measure performance (threshold speed — $\sigma_v$ — for a given number of targets $n_t$, when spacing is fixed, $\sigma_x = 4$, and the total number of objects is also fixed $n = 16$). The characteristic tradeoff between the number of targets, and the speed with which they may be tracked is clearly evident. Thus, while many results in MOT arise as consequences of the information available for the computational task, the speed-number tradeoff seems to be the result of a flexibly-allocated resource such as memory or attention.

## 4   Conclusions

We investigated what limitations are responsible for human failures in multiple object tracking tasks. Are such limitations discrete (like a fixed number of objects) or continuous (like memory)? Are they flexible with task (cognitive resources such as memory and attention), or are they task-independent (like perceptual noise)?

We modified a Bayes-optimal tracking solution for typical MOT experiments and implemented this solution using a Rao-Blackwellized particle filter. Using novel behavioral experiments inspired by the model, we showed that this ideal observer exhibits many of the classic phenomena in multiple object tracking given only perceptual uncertainty (a continuous, task-independent source of limitation). Just as for human observers, tracking in our model is harder when objects move faster or are closer together; inertia information is available, but may not be used; and objects can be tracked in features as well as space. However, effects of the number of objects tracked do not arise from perceptual uncertainty alone. To account for the tradeoff between the number of objects tracked and their speed, a task-dependent resource must be introduced – we introduce this resource as a memory constraint, but it may well be attentional gain.

Although the dichotomy of flexible, cognitive resources and task-independent, low-level uncertainty is a convenient distinction to start our analysis, it is misleading. When engaging in any real world task this distinction is blurred: people will use whatever resources they have to facilitate performance; even perceptual uncertainty as basic as the resolution of the retina becomes a flexible resource when people are allowed to move their eyes (they were not allowed to do so in our experiments). Connecting resource limitations measured in controlled experiments to human performance in the real world requires that we address not only what the structure of the task may be, but also how human agents allocate resources to accomplish this task. Here we have shown that a computational model of the multiple object tracking task can unify a large set of experimental findings on human object tracking, and most importantly, determine how these experimental findings map onto cognitive limitations. Because our findings implicate a flexible cognitive resource, the next necessary step is to investigate how people allocate such a resource, and this question will be pursued in future work.

**Acknowledgments:** This work was supported by ONR MURI: Complex Learning and Skill Transfer with Video Games N00014-07-1-0937 (PI: Daphne Bavelier); NDSEG fellowship to EV and NSF DRMS Dissertation grant to EV.

## Footnotes

[1]The correlation of this method with participants' objective tracking performance was validated by [1].

[2]We used the Weibull cumulative density as our psychometric function $p = 1 - \exp(x/x_{crit})^s$, where $x$ is the stimulus dimension which, which covaries positively with performance (either $\sigma_x$, or $1/\sigma_v$), $x_{crit}$ is the location term, and $s$ is the scale, or slope, parameter. We obtained the MAP estimate of both parameters of the Weibull density function, and predicted the model's 85% threshold (blue plane in Figure 2) from the inverse of the psychometric function: $x = -x_{crit} \ln(1-p)^{1/s}$.

[3]One might suppose that limiting the number of particles used for tracking as in [4] and [14], might be a likely resource capacity; however, in object tracking, having more particles produces a benefit only insofar as future observations might disambiguate previous inferences. In multiple object tracking with uniform dots (as is the case in most human experiments) once objects have been mis-associated, no future observations can provide evidence of a mistake having been made in the past; and as such, having additional particles to keep track of low-probability data associations carries no benefit.

# References

[1] G. Alvarez and S. Franconeri. How many objects can you attentively track?: Evidence for a resource-limited tracking mechanism. *Journal of Vision*, 7(13):1–10, 2007.

[2] P. Bays and M. Husain. Dynamic shifts of limited working memory resources in human vision. *Science*, 321(5890):851, 2008.

[3] E. Blaser, Z. Pylyshyn, and A. Holcombe. Tracking an object through feature space. *Nature*, 408(6809):196 – 199, 2000.

[4] S. Brown and M. Steyvers. Detecting and predicting changes. *Cognitive Psychology*, 58:49–67, 2008.

[5] M. Carrasco and K. Frieder. Cortical magnification neutralizes the eccentricity effect in visual search. *Vision Research*, 37(1):63–82, 1997.

[6] A. Doucet, N. de Freitas, K. Murphy, and S. Russell. Rao-Blackwellised particle filtering for dynamic Bayesian networks. In *Proceedings of Uncertainty in AI*, volume 00, 2000.

[7] J. Feldman and P. Tremoulet. Individuation of visual objects over time. *Cognition*, 99:131–165, 2006.

[8] D. E. Fencsik, J. Urrea, S. S. Place, J. M. Wolfe, and T. S. Horowitz. Velocity cues improve visual search and multiple object tracking. *Visual Cognition*, 14:92–95, 2006.

[9] S. Franconeri, J. Lin, Z. Pylyshyn, B. Fisher, and J. Enns. Evidence against a speed limit in multiple object tracking. *Psychonomic Bulletin & Review*, 15:802–808, 2008.

[10] F. Gustafsson, F. Gunnarsson, N. Bergman, U. Forssell, J. Jansson, R. Karlsson, and P. Nordlund. Particle filters for positioning, navigation, and tracking. In *IEEE Transactions on Signal Processing*, volume 50, 2002.

[11] W. James. *The Principles of Psychology*. Harvard University Press, Cambridge, 1890.

[12] R. Kalman. A new approach to linear filtering and prediction problems. *J. of Basic Engineering*, 82D:35–45, 1960.

[13] B. P. Keane and Z. W. Pylyshyn. Is motion extrapolation employed in multiple object tracking? Tracking as a low-level non-predictive function. *Cognitive Psychology*, 52:346 – 368, 2006.

[14] R. Levy, F. Reali, and T. Griffiths. Modeling the effects of memory on human online sentence processing with particle filters. In *Advances in Neural Information Processing Systems*, volume 21, 2009.

[15] T. Makovski and Y. Jiang. Feature binding in attentive tracking of distinct objects. *Visual Cognition*, 17:180 – 194, 2009.

[16] Z. W. Pylyshyn and R. W. Storm. Tracking multiple independent targets: Evidence for a parallel tracking mechanism. *Spatial Vision*, 3:179–197, 1988.

[17] R. Snowden and O. Braddick. The temporal integration and resolution of velocity signals. *Vision Research*, 31(5):907–914, 1991.

[18] R. Streit and Luginbuhl. Probabilistic multi-hypothesis tracking. *Technical report 10428, NUWC, Newport, Rhode Island, USA*, 1995.

[19] S. P. Tripathy and B. T. Barrett. Severe loss of positional information when detecting deviations in multiple trajectories. *Journal of Vision*, 4(12):1020 – 1043, 2004.

